# A Mixture of Experts Classifier with Learning Based on Both Labelled and Unlabelled Data

David J. Miller and Hasan S. Uyar
Department of Electrical Engineering
The Pennsylvania State University
University Park, Pa. 16802
miller@perseus.ee.psu.edu

## Abstract

We address statistical classifier design given a mixed training set consisting of a small labelled feature set and a (generally larger) set of *unlabelled* features. This situation arises, e.g., for medical images, where although training features may be plentiful, expensive expertise is required to extract their class labels. We propose a classifier structure and learning algorithm that make effective use of unlabelled data to improve performance. The learning is based on maximization of the total data likelihood, i.e. over both the labelled and unlabelled data subsets. Two *distinct* EM learning algorithms are proposed, differing in the EM formalism applied for unlabelled data. The classifier, based on a joint probability model for features and labels, is a "mixture of experts" structure that is equivalent to the radial basis function (RBF) classifier, but unlike RBFs, is amenable to likelihood-based training. The scope of application for the new method is greatly extended by the observation that test data, or any new data to classify, is in fact additional, *unlabelled data* – thus, a combined learning/classification operation – much akin to what is done in image segmentation – can be invoked whenever there is new data to classify. Experiments with data sets from the UC Irvine database demonstrate that the new learning algorithms and structure achieve substantial performance gains over alternative approaches.

## 1 Introduction

Statistical classifier design is fundamentally a supervised learning problem, wherein a decision function, mapping an input feature vector to an output class label, is learned based on representative (feature,class label) training pairs. While a variety of classifier structures and associated learning algorithms have been developed, a common element of nearly all approaches is the assumption that class labels are

known for each feature vector used for training. This is certainly true of neural networks such as multilayer perceptrons and radial basis functions (RBFs), for which classification is usually viewed as function approximation, with the networks trained to minimize the squared distance to target class values. Knowledge of class labels is also required for parametric classifiers such as mixture of Gaussian classifiers, for which learning typically involves dividing the training data into subsets by class and then using maximum likelihood estimation (MLE) to separately learn each class density. While labelled training data may be plentiful for some applications, for others, such as remote sensing and medical imaging, the training set is in principle vast but the size of the labelled subset may be inadequate. The difficulty in obtaining class labels may arise due to limited knowledge or limited resources, as expensive expertise is often required to derive class labels for features. In this work, we address classifier design under these conditions, i.e. the training set $\mathcal{X}$ is assumed to consist of two subsets, $\mathcal{X} = \{\mathcal{X}_l, \mathcal{X}_u\}$, where $\mathcal{X}_l = \{(x_1, c_1), (x_2, c_2), \ldots, (x_{N_l}, c_{N_l})\}$ is the labelled subset and $\mathcal{X}_u = \{x_{N_l+1}, \ldots, x_N\}$ is the unlabelled subset[1]. Here, $x_i \in \mathcal{R}^k$ is the feature vector and $c_i \in \mathcal{I}$ is the class label from the label set $\mathcal{I} = \{1, 2, \ldots, N_c\}$.

The practical significance of this mixed training problem was recognized in (Lippmann 1989). However, despite this realization, there has been surprisingly little work done on this problem. One likely reason is that it does not appear possible to incorporate unlabelled data directly within conventional supervised learning methods such as back propagation. For these methods, unlabelled features must either be discarded or preprocessed in a suboptimal, heuristic fashion to obtain class label estimates. We also note the existence of work which is less than optimistic concerning the value of unlabelled data for classification (Castelli and Cover 1994). However, (Shashahani and Landgrebe 1994) found that unlabelled data could be used effectively in label-deficient situations. While we build on their work, as well as on our own previous work (Miller and Uyar 1996), our approach differs from (Shashahani and Landgrebe 1994) in several important respects. First, we suggest a more powerful mixture-based probability model with an associated classifier structure that has been shown to be equivalent to the RBF classifier (Miller 1996). The practical significance of this equivalence is that unlike RBFs, which are trained in a conventional supervised fashion, the RBF-equivalent mixture model is naturally suited for statistical training (MLE). The statistical framework is the key to incorporating unlabelled data in the learning. A second departure from prior work is the choice of learning criterion. We maximize the joint data likelihood and suggest two *distinct* EM algorithms for this purpose, whereas the conditional likelihood was considered in (Shashahani and Landgrebe 1994). We have found that our approach achieves superior results. A final novel contribution is a considerable expansion of the range of situations for which the mixed training paradigm can be applied. This is made possible by the realization that test data or new data to classify can *also* be viewed as an unlabelled set, available for "training". This notion will be clarified in the sequel.

## 2   Unlabelled Data and Classification

Here we briefly provide some intuitive motivation for the use of unlabelled data. Suppose, not very restrictively, that the data is well-modelled by a mixture density,

in the following way. The feature vectors are generated according to the density $f(x/\theta) = \sum\limits_{l=1}^{L} \alpha_l f(x/\theta_l)$, where $f(x/\theta_l)$ is one of $L$ component densities, with non-negative mixing parameters $\alpha_l$, such that $\sum\limits_{l=1}^{L} \alpha_l = 1$. Here, $\theta_l$ is the set of parameters specifying the component density, with $\theta = \{\theta_l\}$. The class labels are also viewed as *random* quantities and are assumed chosen conditioned on the selected mixture component $m_i \in \{1, 2, \ldots, L\}$ and possibly on the feature value, i.e. according to the probabilities $P[c_i/x_i, m_i]$ [2]. Thus, the data pairs are assumed generated by selecting, in order, the mixture component, the feature value, and the class label, with each selection depending in general on preceding ones. The optimal classification rule for this model is the maximum a posteriori rule:

$$S(x) = \arg \max_k \sum_j P[c_i = k/m_i = j, x_i] P[m_i = j/x_i], \qquad (1)$$

where $P[m_i = j/x_i] = \dfrac{\alpha_j f(x_i/\theta_j)}{\sum\limits_{l=1}^{L} \alpha_l f(x_i/\theta_l)}$, and where $S(x)$ is a selector function with range in $\mathcal{I}$. Since this rule is based on the a posteriori class probabilities, one can argue that learning should focus solely on estimating these probabilities. However, if the classifier truly implements (1), then implicitly it has been assumed that the estimated mixture density accurately models the feature vectors. If this is not true, then presumably estimates of the a posteriori probabilities will also be affected. This suggests that *even in the absence of class labels*, the feature vectors can be used to better learn a posteriori probabilities via improved estimation of the mixture-based feature density. A commonly used measure of mixture density accuracy is the data *likelihood.*

## 3 Joint Likelihood Maximization for a Mixtures of Experts Classifier

The previous section basically argues for a learning approach that uses labelled data to directly estimate a posteriori probabilities and unlabelled data to estimate the feature density. A criterion which essentially fulfills these objectives is the joint data likelihood, computed over both the labelled and unlabelled data subsets. Given our model, the joint data log-likelihood is written in the form

$$\log L = \sum_{x_i \in \mathcal{X}_u} \log \sum_{l=1}^{L} \alpha_l f(x_i/\theta_l) + \sum_{x_i \in \mathcal{X}_l} \log \sum_{l=1}^{L} \alpha_l P[c_i/x_i, m_i = l] f(x_i/\theta_l). \quad (2)$$

This objective function consists of a "supervised" term based on $\mathcal{X}_l$ and an "unsupervised" term based on $\mathcal{X}_u$. The joint data likelihood was previously considered in a learning context in (Xu et al. 1995). However, there the primary justification was simplification of the learning algorithm in order to allow parameter estimation based on fixed point iterations rather than gradient descent. Here, the joint likelihood allows the inclusion of unlabelled samples in the learning. We next consider two special cases of the probability model described until now.

The "partitioned" mixture (PM) model: This is the previously mentioned case where mixture components are "hard-partitioned" to classes (Shashahani and Landgrebe 1994). This is written $M_j \in C_k$, where $M_j$ denotes mixture component $j$ and $C_k$ is the subset of components owned by class $k$. The posterior probabilities have the form

$$P[c_i = k/x] = \frac{\sum\limits_{j:M_j \in C_k} \alpha_j f(x/\theta_j)}{\sum\limits_{l=1}^{L} \alpha_l f(x/\theta_l)}. \tag{3}$$

The generalized mixture (GM) model: The form of the posterior for each mixture component is now $P[c_i/m_i, x_i] = P[c_i/m_i] \equiv \beta_{c_i/m_i}$, i.e., it is independent of the feature value. The overall posterior probability takes the form

$$P[c_i/x_i] = \sum_j \left( \frac{\alpha_j f(x_i/\theta_j)}{\sum_l \alpha_l f(x_i/\theta_l)} \right) \beta_{c_i|j}. \tag{4}$$

This model was introduced in (Miller and Uyar 1996) and was shown there to lead to performance improvement over the PM model. Note that the probabilities have a "mixture of experts" structure, where the "gating units" are the probabilities $P[m_i = j|x_i]$ (in parentheses), and with the "expert" for component $j$ just the probability $\beta_{c_i|j}$. Elsewhere (Miller 1996), it has been shown that the associated classifier decision function is in fact equivalent to that of an RBF classifier (Moody and Darken 1989). Thus, we suggest a probability model equivalent to a widely used neural network classifier, but with the advantage that, unlike the standard RBF, the RBF-equivalent probability model is amenable to statistical training, and hence to the incorporation of unlabelled data in the learning. Note that more powerful models $P[c_i|m_i, x_i]$ that do condition on $x_i$ are also possible. However, such models will require many more parameters which will likely hurt generalization performance, especially in a label-deficient learning context. Interestingly, for the mixed training problem, there are two Expectation-Maximization (EM) (Dempster et al. 1977) formulations that can be applied to maximize the likelihood associated with a given probability model. These two formulations lead to *distinct* methods that take different learning "trajectories", although both ascend in the data likelihood. The difference between the formulations lies in how the *"incomplete"* and *"complete"* data elements are defined within the EM framework. We will develop these two approaches for the suggested GM model.

EM-I (No class labels assumed): Distinct data interpretations are given for $\mathcal{X}_l$ and $\mathcal{X}_u$. In this case, for $\mathcal{X}_u$, the *incomplete data* consists of the features $\{x_i\}$ and the *complete data* consists of $\{(x_i, m_i)\}$. For $\mathcal{X}_l$, the *incomplete data* consists of $\{(x_i, c_i)\}$, with the *complete data* now the triple $\{(x_i, c_i, m_i)\}$. To clarify, in this case *mixture* labels are viewed as the sole missing data elements, for $\mathcal{X}_u$ as well as for $\mathcal{X}_l$. Thus, in effect class labels are not even postulated to exist for $\mathcal{X}_u$.

EM-II (Class labels assumed): The definitions for $\mathcal{X}_l$ are the same as before. However, for $\mathcal{X}_u$, the *complete data* now consists of the triple $\{(x_i, c_i, m_i)\}$, i.e. class labels are also assumed missing for $\mathcal{X}_u$.

For Gaussian components, we have $\theta_l = \{\mu_l, \Sigma_l\}$, with $\mu_l$ the mean vector and $\Sigma_l$ the covariance matrix. For EM-I, the resulting fixed point iterations for updating the parameters are:

$$\mu_j^{(t+1)} = \frac{1}{N\alpha_j^{(t)}} \left( \sum_{x_i \in \mathcal{X}_l} x_i P[m_i = j/x_i, c_i, \theta^{(t)}] + \sum_{x_i \in \mathcal{X}_u} x_i P[m_i = j/x_i, \theta^{(t)}] \right)$$

$$\Sigma_j^{(t+1)} = \frac{1}{N\alpha_j^{(t)}} \left( \sum_{x_i \in \mathcal{X}_l} S_{ij}^{(t)} P[m_i = j/x_i, c_i, \theta^{(t)}] + \sum_{x_i \in \mathcal{X}_u} S_{ij}^{(t)} P[m_i = j/x_i, \theta^{(t)}] \right)$$

$$\alpha_j^{(t+1)} = \frac{1}{N} \left( \sum_{x_i \in \mathcal{X}_l} P[m_i = j/x_i, c_i, \theta^{(t)}] + \sum_{x_i \in \mathcal{X}_u} P[m_i = j/x_i, \theta^{(t)}] \right) \quad \forall j$$

$$\beta_{k/j}^{(t+1)} = \frac{\sum\limits_{x_i \in \mathcal{X}_l \cap c_i = k} P[m_i = j/x_i, c_i, \theta^{(t)}]}{\sum\limits_{x_i \in \mathcal{X}_l} P[m_i = j/x_i, c_i, \theta^{(t)}]} \quad \forall k, j \tag{5}$$

Here, $S_{ij}^{(t)} \equiv (x_i - \mu_j^{(t)})(x_i - \mu_j^{(t)})^T$. New parameters are computed at iteration $t+1$ based on their values at iteration $t$. In these equations, $P[m_i = j/x_i, c_i, \theta^{(t)}] = \frac{\alpha_j^{(t)}\beta_{c_i/j}^{(t)}f(x_i/\theta_j^{(t)})}{\sum\limits_m \alpha_m^{(t)}\beta_{c_i/m}^{(t)}f(x_i/\theta_m^{(t)})}$ and $P[m_i = j/x_i, \theta^{(t)}] = \frac{\alpha_j^{(t)}f(x_i/\theta_j^{(t)})}{\sum\limits_{n=1}^{M} \alpha_n^{(t)}f(x_i/\theta_n^{(t)})}$. For EM-II, it can be

shown that the resulting re-estimation equations are identical to those in (5) *except* regarding the parameters $\{\beta_{k/j}\}$. The updates for these parameters now take the form

$$\beta_{k/j}^{(t+1)} = \frac{1}{N\alpha_j^{(t)}} \left( \sum_{x_i \in \mathcal{X}_l \cap c_i = k} P[m_i = j/x_i, c_i, \theta^{(t)}] + \sum_{x_i \in \mathcal{X}_u} P[m_i = j, c_i = k/x_i, \theta^{(t)}] \right)$$

Here, we identify $P[m_i = j, c_i = k/x_i, \theta^{(t)}] = \frac{\alpha_j^{(t)}\beta_{k/j}^{(t)}f(x_i/\theta_j^{(t)})}{\sum\limits_m \alpha_m^{(t)}f(x_i/\theta_m^{(t)})}$. In this formulation,

joint probabilities for class and mixture labels are computed for data in $\mathcal{X}_u$ and used in the estimation of $\{\beta_{k/j}\}$, whereas in the previous formulation $\{\beta_{k/j}\}$ are updated solely on the basis of $\mathcal{X}_l$. While this does appear to be a significant qualitative difference between the two methods, both do ascend in $\log L$, and in practice we have found that they achieve comparable performance.

## 4   Combined Learning and Classification

The range of application for mixed training is greatly extended by the following observation: *test data (with labels withheld), or for that matter, any new batch of data to be classified, can be viewed as a new, unlabelled data set.* Hence, this new data can be taken to be $\mathcal{X}_u$ and used for learning (based on EM-I or EM-II) prior to its classification. What we are suggesting is a combined learning/classification operation that can be applied whenever there is a new batch of data to classify. In the usual supervised learning setting, there is a clear division between the learning and classification (use) phases. In this setting, modification of the classifier for *new* data is not possible (because the data is unlabelled), while for test data such modification is a form of "cheating". However, in our suggested scheme, this learning for unlabelled data is viewed simply as part of the classification operation. This is analogous to image segmentation, wherein we have a common energy function that is minimized for each new image to be segmented. Each such minimization determines a model local to the image *and* a segmentation for the image. Our "segmentation" is just classification, with $\log L$ playing the role of the energy function. It may consist of one term which is always fixed (based on a given labelled training set) and one term which is modified based on each new batch of unlabelled data to classify. We can envision several distinct learning contexts where this scheme can

be used, as well as different ways of realizing the combined learning/classification operation[3] One use is in classification of an image/speech archive, where each image/speaker segment is a separate data "batch". Each batch to classify can be used as an unlabelled "training" set, either in concert with a representative labelled data set, or to modify a design based on such a set[4]. Effectively, this scheme would adapt the classifier to each new data batch. A second application is supervised learning wherein the *total* amount of data is fixed. Here, we need to divide the data into training and test sets with the conflicting goals of i) achieving a good design and ii) accurately measuring generalization performance. Combined learning and classification can be used here to mitigate the loss in performance associated with the choice of a large test set. More generally, our scheme can be used effectively in any setting where the new data to classify is either a) sizable or b) innovative relative to the existing training set.

## 5   Experimental Results

Figure 1a shows results for the 40-dimensional, 3-class *waveform-+noise* data set from the UC Irvine database. The 5000 data pairs were split into equal-size training and test sets. Performance curves were obtained by varying the amount of labelled training data. For each choice of $N_l$, various learning approaches produced 6 solutions based on random parameter initialization, for each of 7 different labelled subset realizations. The test set performance was then averaged over these 42 "trials". All schemes used $L = 12$ components. DA-RBF (Miller et al. 1996) is a deterministic annealing method for RBF classifiers that has been found to achieve very good results, when given adequate training data[5]. However, this supervised learning method is forced to discard unlabelled data, which severely handicaps its performance relative to EM-I, especially for small $N_l$, where the difference is substantial. TEM-I and TEM-II are results for the EM methods (both I and II) in combined learning and classification mode, i.e., where the 2500 test vectors were also used as part of $\mathcal{X}_u$. As seen in the figure, this leads to additional, significant performance gains for small $N_l$. Note also that performance of the two EM methods is comparable. Figure 1b shows results of similar experiments performed on 6-class satellite imagery data (*sat*), also from the UC Irvine database. For this set, the feature dimension is 36, and we chose $L = 18$ components. Here we compared EM-I with the method suggested in (Shashahani and Landgrebe 1994) (SL), based on the PM model. EM-I is seen to achieve substantial performance gains over this alternative learning approach. Note also that the EM-I performance is nearly constant, over the entire range of $N_l$.

Future work will investigate practical applications of combined learning and classification, as well as variations on this scheme which we have only briefly outlined. Moreover, we will investigate possible extensions of the methods described here for the regression problem.

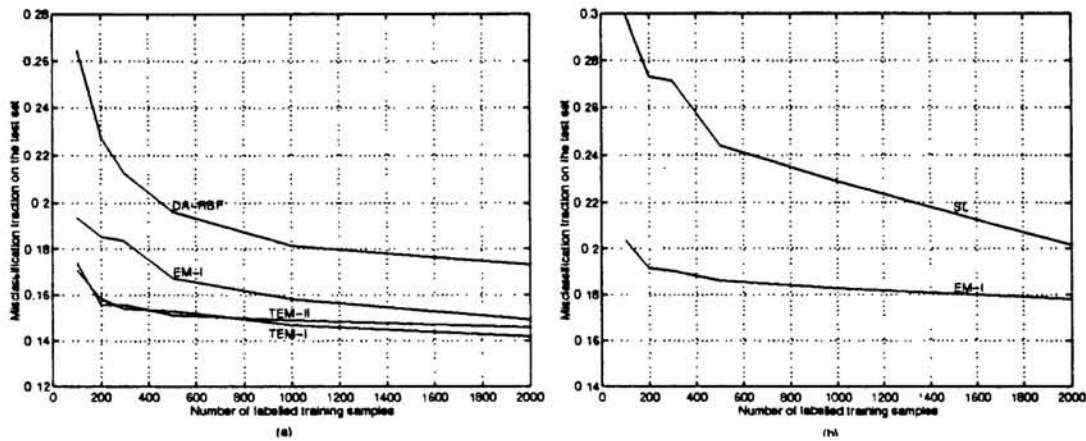

## Acknowledgements

This work was supported in part by National Science Foundation Career Award IRI-9624870.

## Footnotes

[1]This problem can be viewed as a type of "missing data" problem, wherein the missing items are class labels. As such, it is related to, albeit distinct from supervised learning involving missing and/or noisy *feature* components, addressed in (Ghahramani and Jordan 1995),(Tresp et al. 1995).

[2] The usual assumption made is that components are "hard-partitioned", in a deterministic fashion, to classes. Our random model includes the "partitioned" one as a special case. We have generally found this model to be more powerful than the "partitioned" one (Miller Uyar 1996).

[3]The image segmentation analogy in fact suggests an alternative scheme where we perform joint likelihood maximization over both the model parameters *and the "hard"*, *missing class labels*. This approach, which is analogous to segmentation methods such as ICM, would encapsulate the classification operation directly within the learning. Such a scheme will be investigated in future work.

[4]Note that if the classifier is simply modified based on $\mathcal{X}_u$, EM-I will not need to update $\{\beta_{k|j}\}$, while EM-II must update the entire model.

[5]We assumed the same number of basis functions as mixture components. Also, for the DA design, there was only one initialization, since DA is roughly insensitive to this choice.

## References

V. Castelli and T. M. Cover. On the exponential value of labeled samples. *Pattern Recognition Letters*, 16:105–111, 1995.

A.P. Dempster, N.M. Laird, and D.B. Rubin. Maximum-likelihood from incomplete data via the EM algorithm. *Journal of the Roy. Stat. Soc., Ser. B*, 39:1–38, 1977.

Z. Ghahramani and M. I. Jordan. Supervised learning from incomplete data via an EM approach. In *Neural Information Processing Systems 6*, 120-127, 1994.

M. I. Jordan and R. A. Jacobs. Hierarchical mixtures of experts and the EM algorithm. *Neural Computation*, 6:181–214, 1994.

R. P. Lippmann. Pattern classification using neural networks. *IEEE Communications Magazine*, 27, 47-64, 1989.

D. J. Miller, A. Rao, K. Rose, and A. Gersho. A global optimization method for statistical classifier design. *IEEE Transactions on Signal Processing*, Dec. 1996.

D. J. Miller and H. S. Uyar. A generalized Gaussian mixture classifier with learning based on both labelled and unlabelled data. *Conf. on Info. Sci. and Sys.*, 1996.

D. J. Miller. A mixture model equivalent to the radial basis function classifier. Submitted to *Neural Computation*, 1996.

J. Moody and C. J. Darken. Fast learning in locally-tuned processing units. *Neural Computation*, 1:281–294, 1989.

B. Shashahani and D. Landgrebe. The effect of unlabeled samples in reducing the small sample size problem and mitigating the Hughes phenomenon. *IEEE Transactions on Geoscience and Remote Sensing*, 32:1087–1095, 1994.

V. Tresp, R. Neuneier, and S. Ahmad. Efficient methods for dealing with missing data in supervised learning. In *Neural Information Processing Systems 7*, 689-696,1995.

L. Xu, M. I. Jordan, and G. E. Hinton. An alternative model for mixtures of experts. In *Neural Information Processing Systems 7*, 633-640, 1995.
